# Visual Encoding with Jittering Eyes

**Michele Rucci**[*]
Department of Cognitive and Neural Systems
Boston University
Boston, MA 02215
`rucci@cns.bu.edu`

## Abstract

Under natural viewing conditions, small movements of the eye and body prevent the maintenance of a steady direction of gaze. It is known that stimuli tend to fade when they are stabilized on the retina for several seconds. However, it is unclear whether the physiological self-motion of the retinal image serves a visual purpose during the brief periods of natural visual fixation. This study examines the impact of fixational instability on the statistics of visual input to the retina and on the structure of neural activity in the early visual system. Fixational instability introduces fluctuations in the retinal input signals that, in the presence of natural images, lack spatial correlations. These input fluctuations strongly influence neural activity in a model of the LGN. They decorrelate cell responses, even if the contrast sensitivity functions of simulated cells are not perfectly tuned to counter-balance the power-law spectrum of natural images. A decorrelation of neural activity has been proposed to be beneficial for discarding statistical redundancies in the input signals. Fixational instability might, therefore, contribute to establishing efficient representations of natural stimuli.

## 1   Introduction

Models of the visual system often examine steady-state levels of neural activity during presentations of visual stimuli. It is difficult, however, to envision how such steady-states could occur under natural viewing conditions, given that the projection of the visual scene on the retina is never stationary. Indeed, the physiological instability of visual fixation keeps the retinal image in permanent motion even during the brief periods in between saccades.

Several sources cause this constant jittering of the eye. Fixational eye movements, of which we are not aware, alternate small saccades with periods of drifts, even when subjects are instructed to maintain steady fixation [8]. Following macroscopic redirection of gaze, other small eye movements, such as corrective saccades and post-saccadic drifts, are likely to occur. Furthermore, outside of the controlled conditions of a laboratory, when the head is not constrained by a bite bar, movements of the body, as well as imperfections in the vestibulo-ocular reflex, significantly amplify the motion of the retinal image. In the light of

---

[*]Webpage: www.cns.bu.edu/~rucci

this constant jitter, it is remarkable that the brain is capable of constructing a stable percept, as fixational instability moves the stimulus by an amount that should be clearly visible (see, for example, [7]).

Little is known about the purposes of fixational instability. It is often claimed that small saccades are necessary to refresh neuronal responses and prevent the disappearance of a stationary scene, a claim that has remained controversial given the brief durations of natural visual fixation (reviewed in [16]). Yet, recent theoretical proposals [1, 11] have claimed that fixational instability plays a more central role in the acquisition and neural encoding of visual information than that of simply refreshing neural activity. Consistent with the ideas of these proposals, neurophysiological investigations have shown that fixational eye movements strongly influence the activity of neurons in several areas of the monkey's brain [5, 14, 6]. Furthermore, modeling studies that simulated neural responses during free-viewing suggest that fixational instability profoundly affects the statistics of thalamic [13] and thalamocortical activity [10].

This paper summarizes an alternative theory for the existence of fixational instability. Instead of regarding the jitter of visual fixation as necessary for *refreshing* neuronal responses, it is argued that the self-motion of the retinal image is essential for properly *structuring* neural activity in the early visual system into a format that is suitable for processing at later stages. It is proposed that fixational instability is part of a strategy of acquisition of visual information that enables compact visual representations in the presence of natural visual input.

## 2   Neural decorrelation and fixational instability

It is a long-standing proposal that an important function of early visual processing is the removal of part of the redundancy that characterizes natural visual input [3]. Less redundant signals enable more compact representations, in which the same amount of information can be represented by smaller neuronal ensembles. While several methods exist for eliminating input redundancies, a possible approach is the removal of pairwise correlations between the intensity values of nearby pixels [2]. Elimination of these spatial correlations allows efficient representations in which neuronal responses tend to be less statistically dependent.

According to the theory described in this paper, fixational instability contributes to decorrelating the responses of cells in the retina and the LGN during viewing of natural scenes. This theory is based on two factors, which are described separately in the following sections. The first component, analyzed in Section 2.1, is the spatially uncorrelated input signal that occurs when natural scenes are scanned by jittering eyes. The second factor is an amplification of this spatially uncorrelated input, which is mediated by cell response characteristics. Section 2.2 examines the interaction between the dynamics of fixational instability and the temporal characteristics of neurons in the Lateral Geniculate Nucleus (LGN), the main relay of visual information to the cortex.

### 2.1   Influence of fixational instability on visual input

To analyze the effect of fixational instability on the statistics of geniculate activity, it is useful to approximate the input image in a neighborhood of a fixation point $\mathbf{x}_0$ by means of its Taylor series:

$$I(\mathbf{x}) \approx I(\mathbf{x}_0) + \nabla I(\mathbf{x}_0) \cdot (\mathbf{x} - \mathbf{x}_0)^T + \mathbf{o}(|\mathbf{x} - \mathbf{x}_0|^2) \tag{1}$$

If the jittering produced by fixational instability is sufficiently small, high-order derivatives can be neglected, and the input to a location $\mathbf{x}$ on the retina during visual fixation can be approximated by its first-order expansion:

$$S(\mathbf{x}, t) \approx I(\mathbf{x}) + \boldsymbol{\xi}^T(t) \cdot \nabla I(\mathbf{x}) = I(\mathbf{x}) + \tilde{I}(\mathbf{x}, t) \tag{2}$$

where $\boldsymbol{\xi}(t) = [\xi_x(t), \xi_y(t)]$ is the trajectory of the center of gaze during the period of fixation, $t$ is the time elapsed from fixation onset, $I(\mathbf{x})$ is the visual input at $t = 0$, and $\tilde{I}(\mathbf{x}, t) = \frac{\partial I(\mathbf{x})}{\partial x}\xi_x(t) + \frac{\partial I(\mathbf{x})}{\partial y}\xi_y(t)$ is the dynamic fluctuation in the visual input produced by fixational instability.

Eq. 2 allows an analytical estimation of the power spectrum of the signal entering the eye during the self-motion of the retinal image. Since, according to Eq. 2, the retinal input $S(\mathbf{x}, t)$ can be approximated by the sum of two contributions, $I$ and $\tilde{I}$, its power spectrum $R_{SS}$ consists of three terms:

$$R_{SS}(\mathbf{u}, w) \approx R_{II} + R_{\tilde{I}\tilde{I}} + 2R_{I\tilde{I}}$$

where $\mathbf{u}$ and $w$ represent, respectively, spatial and temporal frequency.

Fixational instability can be modeled as an ergodic process with zero mean and uncorrelated components along the two axes, *i.e.*, $\langle\boldsymbol{\xi}\rangle_T = \mathbf{0}$ and $R_{\xi_x\xi_y}(t) = 0$. Although not necessary for the proposed theory, these assumptions simplify our statistical analysis, as $R_{I\tilde{I}}$ is zero, and the power spectrum of the visual input is given by:

$$R_{SS} \approx R_{II} + R_{\tilde{I}\tilde{I}} \tag{3}$$

where $R_{II}$ is the power spectrum of the stimulus, and $R_{\tilde{I}\tilde{I}}$ depends on both the stimulus and fixational instability.

To determine $R_{\tilde{I}\tilde{I}}(\mathbf{u}, w)$, from Eq. 2 follows that

$$\tilde{I}(\mathbf{u}, w) = iu_x I(\mathbf{u})\xi_x(w) + iu_y I(\mathbf{u})\xi_y(w)$$

and under the assumption of uncorrelated motion components, approximating the power spectrum via finite Fourier Transform yields:

$$R_{\tilde{I}\tilde{I}}(\mathbf{u}, w) = \lim_{T \to \infty} < \frac{1}{T}|\tilde{I}_T(\mathbf{u}, w)|^2 >_{\xi, \mathcal{I}} = R_{\xi\xi}(w)R_{II}(\mathbf{u})|\mathbf{u}|^2 \tag{4}$$

where $\tilde{I}_T$ is the Fourier Transform of a signal of duration $T$, and we have assumed identical second-order statistics of retinal image motion along the two Cartesian axes. As shown in Fig. 1 is clear that the presence of the term $\mathbf{u}^2$ in Eq. 4 compensates for the scaling invariance of natural images. That is, since for natural images $R_{II}(\mathbf{u}) \propto \mathbf{u}^{-2}$, the product $R_{II}(\mathbf{u})|\mathbf{u}|^2$ whitens $R_{II}$ by producing a power spectrum $R_{\tilde{I}\tilde{I}}$ that remains virtually constant at all spatial frequencies.

## 2.2   Influence of fixational instability on neural activity

This section analyzes the structure of correlated activity during fixational instability in a model of the LGN. To delineate the important elements of the theory, we consider linear approximations of geniculate responses provided by space-time separable kernels. This assumption greatly simplifies the analysis of levels of correlation. Results are, however, general, and the outcomes of simulations with space-time inseparable kernels and different levels of rectification (the most prominent nonlinear behavior of parvocellular geniculate neurons) can be found in [13, 10].

Mean instantaneous firing rates were estimated on the basis of the convolution between the input $I$ and the cell spatiotemporal kernel $h_\alpha$:

$$\alpha(t) = h_\alpha(\mathbf{x}, t) \star I(\mathbf{x}, t) = \int_0^t \int_{-\infty}^{\infty} \int_{-\infty}^{\infty} h_\alpha(x', y', t')I(x - x', y - y', t - t') \, dx' \, dy' \, dt'$$

where $h_\alpha(\mathbf{x}, t) = g_\alpha(t)f_\alpha(\mathbf{x})$. Kernels were designed on the basis of data from neurophysiological recordings to replicate the responses of parvocellular ON-center cells in the LGN

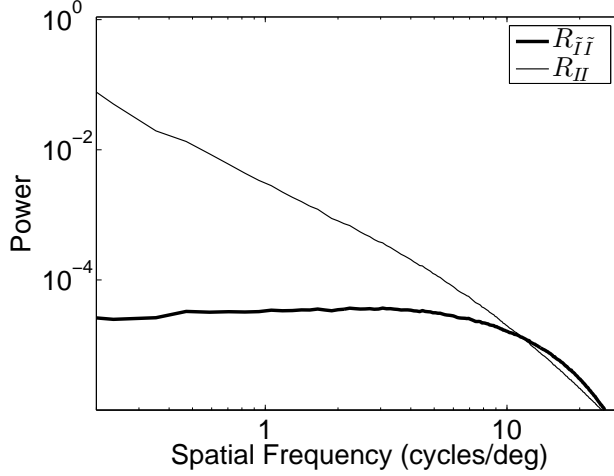

Figure 1: Fixational instability introduces a spatially uncorrelated component in the visual input to the retina during viewing of natural scenes. The graph compares the power spectrum of natural images ($R_{II}$) to the dynamic power spectrum introduced by fixational instability ($R_{\tilde{I}\tilde{I}}$). The two curves represent radial averages evaluated over 15 pictures of natural scenes.

of the macaque. The spatial component $f_\alpha(\mathbf{x})$ was modeled by a standard difference of Gaussian [15]. The temporal kernel $g_\alpha(t)$ possessed a biphasic profile with positive peak at 50 ms, negative peak at 75 ms, and overall duration of less than 200 ms [4].

In this section, levels of correlation in the activity of pairs of geniculate neurons are summarized by the correlation pattern $\hat{c}_{\alpha\alpha}(\mathbf{x})$:

$$\hat{c}_{\alpha\alpha}(\mathbf{x}) = \langle \alpha_{\mathbf{y}}(t)\alpha_{\mathbf{z}}(t)\rangle\Big|_{T,\mathcal{I}} \tag{5}$$

where $\alpha_{\mathbf{y}}(t)$ and $\alpha_{\mathbf{z}}(t)$ are the responses of cells with receptive fields centered at $\mathbf{y}$ and $\mathbf{z}$, and $\mathbf{x} = \mathbf{y} - \mathbf{z}$ is the separation between receptive field centers. The average is evaluated over time $T$ and over a set of stimuli $\mathcal{I}$.

With linear models, $\hat{c}_{\alpha\alpha}(\mathbf{x})$ can be estimated on the basis of the input power spectrum $R_{SS}(\mathbf{u}, w)$:

$$\hat{c}_{\alpha\alpha}(\mathbf{x}) = c_{\alpha\alpha}(\mathbf{x}, t)\Big|_{t=0} \quad \text{and} \quad c_{\alpha\alpha}(\mathbf{x}, t) = \mathcal{F}^{-1}\{R_{\alpha\alpha}\} \tag{6}$$

where $R_{\alpha\alpha} = |H_\alpha|^2 R_{SS}(\mathbf{u}, w)$ is the power spectrum of LGN activity ($H_\alpha(\mathbf{u}, w)$ is the spatiotemporal Fourier transform of the kernel $h_\alpha(\mathbf{x}, t)$), and $\mathcal{F}^{-1}$ represents the inverse Fourier transform operator.

To evaluate $R_{\alpha\alpha}$, substitution of $R_{SS}$ from Eq. 3 and separation of spatial and temporal elements yield:

$$R_{\alpha\alpha} \approx |G_\alpha|^2|F_\alpha|^2 R_{II} + |G_\alpha|^2|F_\alpha|^2 R_{\tilde{I}\tilde{I}} = R_{\alpha\alpha}^S + R_{\alpha\alpha}^D \tag{7}$$

where $F_\alpha(\mathbf{u})$ and $G_\alpha(w)$ represent the Fourier Transforms of the spatial and temporal kernels. Eq. 7 shows that, similar to the retinal input, also the power spectrum of geniculate activity can be approximated by the sum of two separate elements. Only $R_{\alpha\alpha}^D$ depends on fixational instability. The first term, $R_{\alpha\alpha}^S$, is determined by the power spectrum of the

stimulus and the characteristics of geniculate cells but does not depend on the motion of the eye during the acquisition of visual information.

By substituting in Eq. 6 the expression of $R_{\alpha\alpha}$ from Eq. 7, we obtain

$$c_{\alpha\alpha}(\mathbf{x}, t) \approx c_{\alpha\alpha}^{S}(\mathbf{x}, t) + c_{\alpha\alpha}^{D}(\mathbf{x}, t) \tag{8}$$

where

$$c_{\alpha\alpha}^{S}(\mathbf{x}, t) = \mathcal{F}^{-1}\{R_{\alpha\alpha}^{S}(\mathbf{u}, w)\} \text{ and } c_{\alpha\alpha}^{D}(\mathbf{x}, t) = \mathcal{F}^{-1}\{R_{\alpha\alpha}^{D}(\mathbf{u}, w)\}$$

Eq. 8 shows that fixational instability adds the term $c_{\alpha\alpha}^{D}$ to the pattern of correlated activity $c_{\alpha\alpha}^{S}$ that would obtained with presentation of the same set of stimuli without the self-motion of the eye.

With presentation of pictures of natural scenes, $R_{II}(w) = 2\pi\delta(w)$, and the two input signals $R_{\alpha\alpha}^{S}$ and $R_{\alpha\alpha}^{D}$ provide, respectively, a static and a dynamic contribution to the spatiotemporal correlation of geniculate activity. The first term in Eq. 8 gives a correlation pattern:

$$\hat{c}_{\alpha\alpha}^{S}(\mathbf{x}) = k_{S}\mathcal{F}_{S}^{-1}\{|F_{\alpha}|^{2}R_{II}^{S}(\mathbf{u})\} \tag{9}$$

where $k_{S} = |G(0)|^{2}$.

By substituting $R_{\tilde{I}\tilde{I}}$ from Eq. 4, the second term in Eq. 8 gives a correlation pattern:

$$\hat{c}_{\alpha\alpha}^{D}(\mathbf{x}) = k_{D}\mathcal{F}_{\mathcal{S}}^{-1}\{|F_{\alpha}|^{2}R_{II}^{S}(\mathbf{u})|\mathbf{u}|^{2}\} \tag{10}$$

where $k_{D} = \mathcal{F}_{\mathcal{T}}^{-1}\{|G_{\alpha}(w)|^{2}R_{\xi\xi}(w)\}\big|_{t=0}$ is a constant given by the temporal dynamics of cell response and fixational instability. $\mathcal{F}_{T}^{-1}$ and $\mathcal{F}_{S}^{-1}$ indicate the operations of inverse Fourier Transform in time and space.

To summarize, during the physiological instability of visual fixation, the structure of correlated activity in a linear model of the LGN is given by the superposition of two spatial terms, each of them weighted by a coefficient ($k_{S}$ and $k_{D}$) that depends on dynamics:

$$\hat{c}_{\alpha\alpha}(\mathbf{x}) = k_{S}\mathcal{F}_{\mathcal{S}}^{-1}\{(|F_{\alpha}|^{2}R_{II}^{S}(\mathbf{u})\} + k_{D}\mathcal{F}_{\mathcal{S}}^{-1}\{|F_{\alpha}|^{2}R_{II}^{S}(\mathbf{u})|\mathbf{u}|^{2}\} \tag{11}$$

Whereas the stimulus contributes to the structure of correlated activity by means of the power spectrum $R_{II}^{S}$, the contribution introduced by fixational instability depends on $R_{\tilde{I}\tilde{I}}^{S}$, a signal that discards the broad correlation of natural images. Since in natural images, most power is concentrated at low spatial frequencies, the uncorrelated fluctuations in the input signals generated by fixational instability have small amplitudes. That is, $R_{II}^{D}$ provides less power than $R_{\tilde{I}\tilde{I}}^{S}$. However, geniculate cells tend to respond more strongly to changing stimuli than stationary ones, and $k_{D}$ is larger than $k_{S}$. Therefore, the small input modulations introduced by fixational instability are amplified by the dynamics of geniculate cells.

Fig. 2 shows the structure of correlated activity in the model when images of natural scenes are examined in the presence of fixational instability. In this example, fixational instability was assumed to possess Gaussian temporal correlation, $R_{\xi\xi}(w)$, with standard deviation $\sigma_{T} = 22$ ms and amplitude $\sigma_{S} = 12$ arcmin. In addition to the total pattern of correlation given by Eq. 11, Fig. 2 also shows the patterns of correlation produced by the two components $\hat{c}_{\alpha\alpha}^{S}$ and $\hat{c}_{\alpha\alpha}^{D}$. Whereas $\hat{c}_{\alpha\alpha}^{S}$ was strongly influenced by the broad spatial correlations of natural images, $\hat{c}_{\alpha\alpha}^{D}$, due to its dependence on the whitened power spectrum $R_{\tilde{I}\tilde{I}}$, was determined exclusively by cell receptive fields. Due to the amplification factor $k_{D}$, $\hat{c}_{\alpha\alpha}^{D}$ provided a stronger contribution than $\hat{c}_{\alpha\alpha}^{S}$ and heavily influenced the global structure of correlated activity.

To examine the relative influence of the two terms $\hat{c}_{\alpha\alpha}^{S}$ and $\hat{c}_{\alpha\alpha}^{D}$ on the structure of correlated activity, Fig. 3 shows their ratio at separation zero, $\rho_{DS} = \hat{c}_{\alpha\alpha}^{D}(0)/\hat{c}_{\alpha\alpha}^{S}(0)$, with

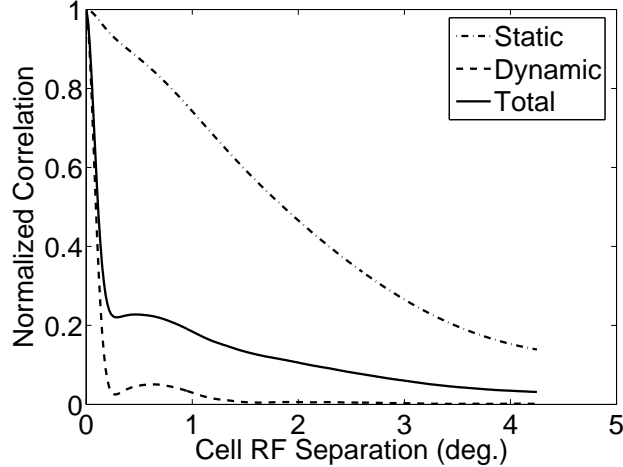

Figure 2: Patterns of correlation obtained from Eq. 11 when natural images are examined in the presence of fixational instability. The three curves represent the total level of correlation (Total), the correlation $\hat{c}^S_{\alpha\alpha}(\mathbf{x})$ that would be present if the same images were examined in the absence of fixational instability (Static), and the contribution $\hat{c}^D_{\alpha\alpha}(\mathbf{x})$ of fixational instability (Dynamic). Data are radial averages evaluated over pairs of cells with the same separation $\|\mathbf{x}\|$ between their receptive fields.

presentation of natural images and for various parameters of fixational instability. Fig. 3 $(a)$ shows the effect of varying the spatial amplitude of the retinal jitter. In order to remain within the range of validity of the Taylor approximation in Eq. 2, only small amplitude values are considered. As shown by Fig. 3 $(a)$, the larger the instability of visual fixation, the larger the contribution of the dynamic term $\hat{c}^D_{\alpha\alpha}$ with respect to $\hat{c}^S_{\alpha\alpha}$. Except for very small values of $\sigma_S$, $\rho_{DS}$ is larger than one, indicating that $\hat{c}^D_{\alpha\alpha}$ influences the structure of correlated activity more strongly than $\hat{c}^S_{\alpha\alpha}$. Fig. 3 $(b)$ shows the impact of varying $\sigma_T$, which defines the temporal window over which fixational jitter is correlated. Note that $\rho_{DS}$ is a non-monotonic function of $\sigma_T$. For a range of $\sigma_T$ corresponding to intervals shorter than the typical duration of visual fixation, $\hat{c}^D_{\alpha\alpha}$ is significantly larger than $\hat{c}^S_{\alpha\alpha}$. Thus, fixational instability strongly influences correlated activity in the model when it moves the direction of gaze within a range of a few arcmin and is correlated over a fraction of the duration of visual fixation. This range of parameters is consistent with the instability of fixation observed in primates.

## 3   Conclusions

It has been proposed that neurons in the early visual system decorrelate their responses to natural stimuli, an operation that is believed to be beneficial for the encoding of visual information [2]. The original claim, which was based on psychophysical measurements of human contrast sensitivity, relies on an inverse proportionality between the spatial response characteristics of retinal and geniculate neurons and the structure of natural images. However, data from neurophysiological recordings have clearly shown that neurons in the retina and the LGN respond significantly to low spatial frequencies, in a way that is not compatible with the requirements of Atick and Redlich's proposal. During natural viewing, input signals to the retina depend not only on the stimulus, but also on the physiological instability of visual fixation. The results of this study show that when natural scenes are examined

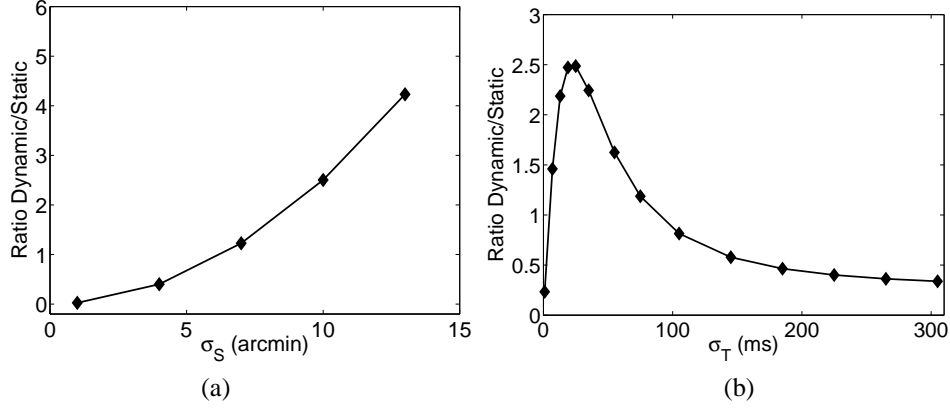

Figure 3: Influence of the characteristics of fixational instability on the patterns of correlated activity during presentation of natural images. The two graphs show the ratio $\rho_{DS}$ between the peaks of the two terms $\hat{c}^D_{\alpha\alpha}$ and $\hat{c}^S_{\alpha\alpha}$ in Eq. 8. Fixational instability was assumed to possess a Gaussian correlation with standard deviation $\sigma_T$ and amplitude $\sigma_S$. (a) Effect of varying $\sigma_S$ ($\sigma_T = 22$ ms). (b) Effect of varying $\sigma_T$ ($\sigma_S = 12$ arcmin).

with jittering eyes, as occurs under natural viewing conditions, fixational instability tends to decorrelate cell responses even if the contrast sensitivity functions of individual neurons do not counterbalance the power spectrum of visual input.

The theory described in this paper relies of two main elements. The first component is the presence of a spatially uncorrelated input signal during presentation of natural visual stimuli ($R_{\tilde{I}\tilde{I}}$ in Eq. 3). This input signal is a direct consequence of the scale invariance of natural images. It is a property of natural images that, although the intensity values of nearby pixels tend to be correlated, changes in intensity around pairs of pixels are uncorrelated. This property is not satisfied by an arbitrary image. In a spatial grating, for example, intensity changes at any two locations are highly correlated. During the instability of visual fixation, neurons receive input from the small regions of the visual field covered by the jittering of their receptive fields. In the presence of natural images, although the inputs to cells with nearby receptive fields are on average correlated, the fluctuations in these input signals produced by fixational instability are not correlated. Fixational instability appears to be tuned to the statistics of natural images, as it introduces a spatially uncorrelated signal only in the presence of visual input with a power spectrum that declines as $u^{-2}$ with spatial frequency.

The second element of the theory is the neuronal amplification of the spatially uncorrelated input signal introduced by the self-motion of the retinal image. This amplification originates from the interaction between the dynamics of fixational instability and the temporal sensitivity of geniculate units. Since $R_{\tilde{I}\tilde{I}}$ attenuates the low spatial frequencies of the stimulus, it tends to possess less power than $R_{II}$. However, in Eq. 11, the contributions of the two input signals are modulated by the multiplicative terms $k_S$ and $k_D$, which depend on the temporal characteristics of cell responses (both $k_S$ and $k_D$) and fixational instability ($k_D$ only). Since geniculate neurons respond more strongly to changing stimuli than to stationary ones, $k_D$ tends to be higher than $k_S$. Correspondingly, in a linear model of the LGN, units are highly sensitive to the uncorrelated fluctuations in the input signals produced by fixational instability.

The theory summarized in this study is consistent with the strong modulations of neural responses observed during fixational eye movements [5, 14, 6], as well as with the results

of recent psychophysical experiments aimed at investigating perceptual influences of fixational instability [12, 9]. It should be observed that, since patterns of correlations were evaluated via Fourier analysis, this study implicitly assumed a steady-state condition of visual fixation. Further work is needed to extend the proposed theory in order to take into account time-varying natural stimuli and the nonstationary regime produced by the occurrence of saccades.

## Acknowledgments

The author thanks Antonino Casile and Gaelle Desbordes for many helpful discussions. This material is based upon work supported by the National Institute of Health under Grant EY15732-01 and the National Science Foundation under Grant CCF-0432104.

## References

[1] E. Ahissar and A. Arieli. Figuring space by time. *Neuron*, 32(2):185–201, 2001.

[2] J. J. Atick and A. Redlich. What does the retina know about natural scenes? *Neural Comp.*, 4:449–572, 1992.

[3] H. B. Barlow. The coding of sensory messages. In W. H. Thorpe and O. L. Zangwill, editors, *Current Problems in Animal Behaviour*, pages 331–360. Cambridge University Press, Cambridge, 1961.

[4] E. A. Benardete and E. Kaplan. Dynamics of primate P retinal ganglion cells: Responses to chromatic and achromatic stimuli. *J. Physiol.*, 519(3):775–790, 1999.

[5] D. A. Leopold and N. K. Logothetis. Microsaccades differentially modulate neural activity in the striate and extrastriate visual cortex. *Exp. Brain. Res.*, 123:341–345, 1998.

[6] S. Martinez-Conde, S. L. Macknik, and D. H. Hubel. The function of bursts of spikes during visual fixation in the awake primate lateral geniculate nucleus and primary visual cortex. *Proc. Natl. Acad. Sci. USA*, 99(21):13920–13925, 2002.

[7] I. Murakami and P. Cavanagh. A jitter after-effect reveals motion-based stabilization of vision. *Nature*, 395(6704):798–801, 1998.

[8] F. Ratliff and L. A. Riggs. Involuntary motions of the eye during monocular fixation. *J. Exp. Psychol.*, 40:687–701, 1950.

[9] M. Rucci and J. Beck. Effects of ISI and flash duration on the identification of briefly flashed stimuli. *Spatial Vision*, 18(2):259–274, 2005.

[10] M. Rucci and A. Casile. Decorrelation of neural activity during fixational instability: Possible implications for the refinement of V1 receptive fields. *Visual Neurosci.*, 21:725–738, 2004.

[11] M. Rucci and A. Casile. Fixational instability and natural image statistics: Implications for early visual representations. *Network: Computation in Neural Systems*, 16(2-3):121–138, 2005.

[12] M. Rucci and G. Desbordes. Contributions of fixational eye movements to the discrimination of briefly presented stimuli. *J. Vision*, 3(11):852–64, 2003.

[13] M. Rucci, G. M. Edelman, and J. Wray. Modeling LGN responses during free-viewing: A possible role of microscopic eye movements in the refinement of cortical orientation selectivity. *J. Neurosci*, 20(12):4708–4720, 2000.

[14] D. M. Snodderly, I. Kagan, and M. Gur. Selective activation of visual cortex neurons by fixational eye movements: Implications for neural coding. *Vis. Neurosci.*, 18:259–277, 2001.

[15] P. D. Spear, R. J. Moore, C. B. Y. Kim, J. T. Xue, and N. Tumosa. Effects of aging on the primate visual system: spatial and temporal processing by lateral geniculate neurons in young adult and old rhesus monkeys. *J. Neurophysiol.*, 72:402–420, 1994.

[16] R.M. Steinman and J.Z. Levinson. The role of eye movements in the detection of contrast and spatial detail. In E. Kowler, editor, *Eye Movements and their Role in Visual and Cognitive Processes*, pages 115–212. Elsevier Science, 1990.
